# Adaptive Back-Propagation in On-Line Learning of Multilayer Networks

**Ansgar H. L. West**[1,2] **and David Saad**[2]
[1]Department of Physics, University of Edinburgh
Edinburgh EH9 3JZ, U.K.
[2]Neural Computing Research Group, University of Aston
Birmingham B4 7ET, U.K.

## Abstract

An adaptive back-propagation algorithm is studied and compared with gradient descent (standard back-propagation) for on-line learning in two-layer neural networks with an arbitrary number of hidden units. Within a statistical mechanics framework, both numerical studies and a rigorous analysis show that the adaptive back-propagation method results in faster training by breaking the symmetry between hidden units more efficiently and by providing faster convergence to optimal generalization than gradient descent.

## 1 INTRODUCTION

Multilayer feedforward perceptrons (MLPs) are widely used in classification and regression applications due to their ability to learn a range of complicated maps [1] from examples. When learning a map $f_0$ from $N$-dimensional inputs $\boldsymbol{\xi}$ to scalars $\zeta$ the parameters $\{\boldsymbol{W}\}$ of the *student* network are adjusted according to some training algorithm so that the map defined by these parameters $f_W$ approximates the *teacher* $f_0$ as close as possible. The resulting performance is measured by the *generalization error* $\epsilon_g$, the average of a suitable error measure $\epsilon$ over all possible inputs $\epsilon_g = \langle \epsilon \rangle_{\boldsymbol{\xi}}$. This error measure is normally defined as the squared distance between the output of the network and the desired output, i.e.,

$$\epsilon = \frac{1}{2}[f_W(\boldsymbol{\xi}) - f_0(\boldsymbol{\xi})]^2. \tag{1}$$

One distinguishes between two learning schemes: *batch learning*, where training algorithms are generally based on minimizing the above error on the whole set of given examples, and *on-line learning*, where single examples are presented serially and the training algorithm adjusts the parameters after the presentation of each

example. We measure the efficiency of these training algorithms by how fast (or whether at all) they converge to an "acceptable" generalization error.

This research has been motivated by recent work [2] investigating an on-line learning scenario of a general two-layer student network trained by gradient descent on a task defined by a teacher network of similar architecture. It has been found that in the early stages of training the student is drawn into a suboptimal symmetric phase, characterized by each student node imitating all teacher nodes with the same degree of success. Although the symmetry between the student nodes is eventually broken and the student converges to the minimal achievable generalization error, the majority of the training time may be spent with the system trapped in the symmetric regime, as one can see in Fig. 1. To investigate possible improvements we introduce an adaptive back-propagation algorithm, which improves the ability of the student to distinguish between hidden nodes of the teacher. We compare its efficiency with that of gradient descent in training two-layer networks following the framework of [2]. In this paper we present numerical studies and a rigorous analysis of both the breaking of the symmetric phase and the convergence to optimal performance. We find that adaptive back-propagation can significantly reduce training time in both regimes by breaking the symmetry between hidden units more efficiently and by providing faster exponential convergence to zero generalization error.

## 2    DERIVATION OF THE DYNAMICAL EQUATIONS

The student network we consider is a soft committee machine [3], consisting of $K$ hidden units which are connected to $N$-dimensional inputs $\boldsymbol{\xi}$ by their weight vectors $\boldsymbol{W} = \{\boldsymbol{W}_i\}$ $(i = 1, \ldots, K)$. All hidden units are connected to the linear output unit by couplings of unit strength and the implemented mapping is therefore $f_W(\boldsymbol{\xi}) = \sum_{i=1}^{K} g(x_i)$, where $x_i = \boldsymbol{W}_i \cdot \boldsymbol{\xi}$ is the activation of hidden unit $i$ and $g(\cdot)$ is a sigmoidal transfer function. The map $f_o$ to be learned is defined by a teacher network of the same architecture except for a possible difference in the number of hidden units $M$ and is defined by the weight vectors $\boldsymbol{B} = \{\boldsymbol{B}_n\}$ $(n = 1, \ldots, M)$. Training examples are of the form $(\boldsymbol{\xi}^\mu, \zeta^\mu)$, where the components of the input vectors $\boldsymbol{\xi}^\mu$ are drawn independently from a zero mean unit variance Gaussian distribution; the outputs are $\zeta^\mu = \sum_{n=1}^{M} g(y_n^\mu)$, where $y_n^\mu = \boldsymbol{B}_n \cdot \boldsymbol{\xi}^\mu$ is the activation of teacher hidden unit $n$.

An on-line training algorithm $\mathcal{A}$ is defined by the update of each weight in response to the presentation of an example $(\boldsymbol{\xi}^\mu, \zeta^\mu)$, which can take the general form $\boldsymbol{W}_i^{\mu+1} = \boldsymbol{W}_i^\mu + \mathcal{A}_i(\{\boldsymbol{\gamma}\}, \boldsymbol{W}^\mu, \boldsymbol{\xi}^\mu, \zeta^\mu)$, where $\{\boldsymbol{\gamma}\}$ defines parameters adjustable by the user. In the case of standard back-propagation, i.e., gradient descent on the error function defined in Eq. (1): $\mathcal{A}_i^{\text{gd}}(\eta, \boldsymbol{W}^\mu, \boldsymbol{\xi}^\mu, \zeta^\mu) = (\eta/N)\delta_i^\mu \boldsymbol{\xi}^\mu$ with

$$\delta_i^\mu = \delta^\mu g'(x_i^\mu) = [\zeta^\mu - f_W(\boldsymbol{\xi}^\mu)] \, g'(x_i^\mu), \qquad (2)$$

where the only user adjustable parameter is the learning rate $\eta$ scaled by $1/N$. One can readily see that the only term that breaks the symmetry between different hidden units is $g'(x_i^\mu)$, i.e., the derivative of the transfer function $g(\cdot)$. The fact that a prolonged symmetric phase can exist indicates that this term is not significantly different over the hidden units for a typical input in the symmetric phase.

The rationale of the adaptive back-propagation algorithm defined below is therefore to alter the $g'$-term, in order to magnify small differences in the activation between hidden units. This can be easily achieved by altering $g'(x_i)$ to $g'(\beta x_i)$, where $\beta$ plays the role of an inverse "temperature". Varying $\beta$ changes the range of hidden unit activations relevant for training, e.g., for $\beta > 1$ learning is more confined to

small activations, when compared to gradient descent ($\beta = 1$). The whole adaptive back-propagation training algorithm is therefore:

$$\mathcal{A}_i^{\text{abp}}(\eta, \beta, \boldsymbol{W}^\mu, \boldsymbol{\xi}^\mu, \zeta^\mu) = \frac{\eta}{N} \delta^\mu g'(\beta x_i^\mu) \boldsymbol{\xi}^\mu = \frac{\eta}{N} \widetilde{\delta_i^\mu} \boldsymbol{\xi}^\mu \qquad (3)$$

with $\delta^\mu$ as in Eq. (2). To compare the adaptive back-propagation algorithm with normal gradient descent, we follow the statistical mechanics calculation in [2]. Here we will only outline the main ideas and present the results of the calculation.

As we are interested in the typical behaviour of our training algorithm we average over all possible instances of the examples $\boldsymbol{\xi}$. We rewrite the update equations (3) in $\boldsymbol{W}_i$ as equations in the order parameters describing the overlaps between student nodes $Q_{ij} = \boldsymbol{W}_i \cdot \boldsymbol{W}_j$, student and teacher nodes $R_{in} = \boldsymbol{W}_i \cdot \boldsymbol{B}_n$ and teacher nodes $T_{nm} = \boldsymbol{B}_n \cdot \boldsymbol{B}_m$. The generalization error $\epsilon_{\text{g}}$, measuring the typical performance, can be expressed in these variables only [2]. The order parameters $Q_{ij}$ and $R_{in}$ are the new dynamical variables, which are self-averaging with respect to the randomness in the training data in the thermodynamic limit ($N \to \infty$). If we interpret the normalized example number $\alpha = \mu/N$ as a continuous time variable, the update equations for the order parameters become first order coupled differential equations

$$\frac{dR_{in}}{d\alpha} = \eta \left\langle \widetilde{\delta}_i y_n \right\rangle_\xi,$$
$$\frac{dQ_{ij}}{d\alpha} = \eta \left\langle \widetilde{\delta}_i x_j + \widetilde{\delta}_j x_i \right\rangle_\xi + \eta^2 \left\langle \widetilde{\delta}_i \widetilde{\delta}_j \right\rangle_\xi. \qquad (4)$$

All the integrals in Eqs. (4) and the generalization error can be calculated explicitly if we choose $g(x) = \text{erf}(x/\sqrt{2})$ as the sigmoidal activation function [2]. The exact form of the resulting dynamical equations for adaptive back-propagation is similar to the equations in [2] and will be presented elsewhere [4]. They can easily be integrated numerically for any number of $K$ student and $M$ teacher hidden units. For the remainder of the paper, we will however focus on the realizable case ($K = M$) and uncorrelated isotropic teachers of unit length $T_{nm} = \delta_{nm}$.

The dynamical evolution of the overlaps $Q_{ij}$ and $R_{in}$ follows from integrating the equations of motion (4) from initial conditions determined by the random initialization of the student weights $\boldsymbol{W}$. Whereas the resulting norms $Q_{ii}$ of the student vector will be order $\mathcal{O}(1)$, the overlaps $Q_{ij}$ between student vectors, and student-teacher vectors $R_{in}$ will be only order $\mathcal{O}(1/\sqrt{N})$. The random initialization of the weights is therefore simulated by initializing the norms $Q_{ii}$ and the overlaps $Q_{ij}$ and $R_{in}$ from uniform distributions in the $[0, 0.5]$ and $[0, 10^{-12}]$ interval respectively.

In Fig. 1 we show the difference of a typical evolution of the overlaps and the generalization error for $\beta = 12$ and $\beta = 1$ (gradient descent) for $K = 3$ and $\eta = 0.01$. In both cases, the student is drawn quickly into a suboptimal symmetric phase, characterized by a finite generalization error (Fig. 1e) and no differentiation between the hidden units of the student: the student norms $Q_{ii}$ and overlaps $Q_{ij}$ are similar (Figs. 1b,1d) and the overlaps of each student node with all teacher nodes $R_{in}$ are nearly identical (Figs. 1a,1c). The student trained by gradient descent (Figs. 1c,1d) is trapped in this unstable suboptimal solution for most of the training time, whereas adaptive back-propagation (Figs. 1a,1b) breaks the symmetry significantly earlier. The convergence phase is characterized by a specialization of the different student nodes and the evolution of the overlap matrices $\mathbf{Q}$ and $\mathbf{R}$ to their optimal value $\mathbf{T}$, except for the permutational symmetry due to the arbitrary labeling of the student nodes. Clearly, the choice $\beta = 12$ is suboptimal in this regime. The student trained with $\beta = 1$ converges faster to zero generalization error (Fig. 1e). In order to optimize $\beta$ seperately for both the symmetric and the convergence phase, we will examine the equations of motions analytically in the following section.

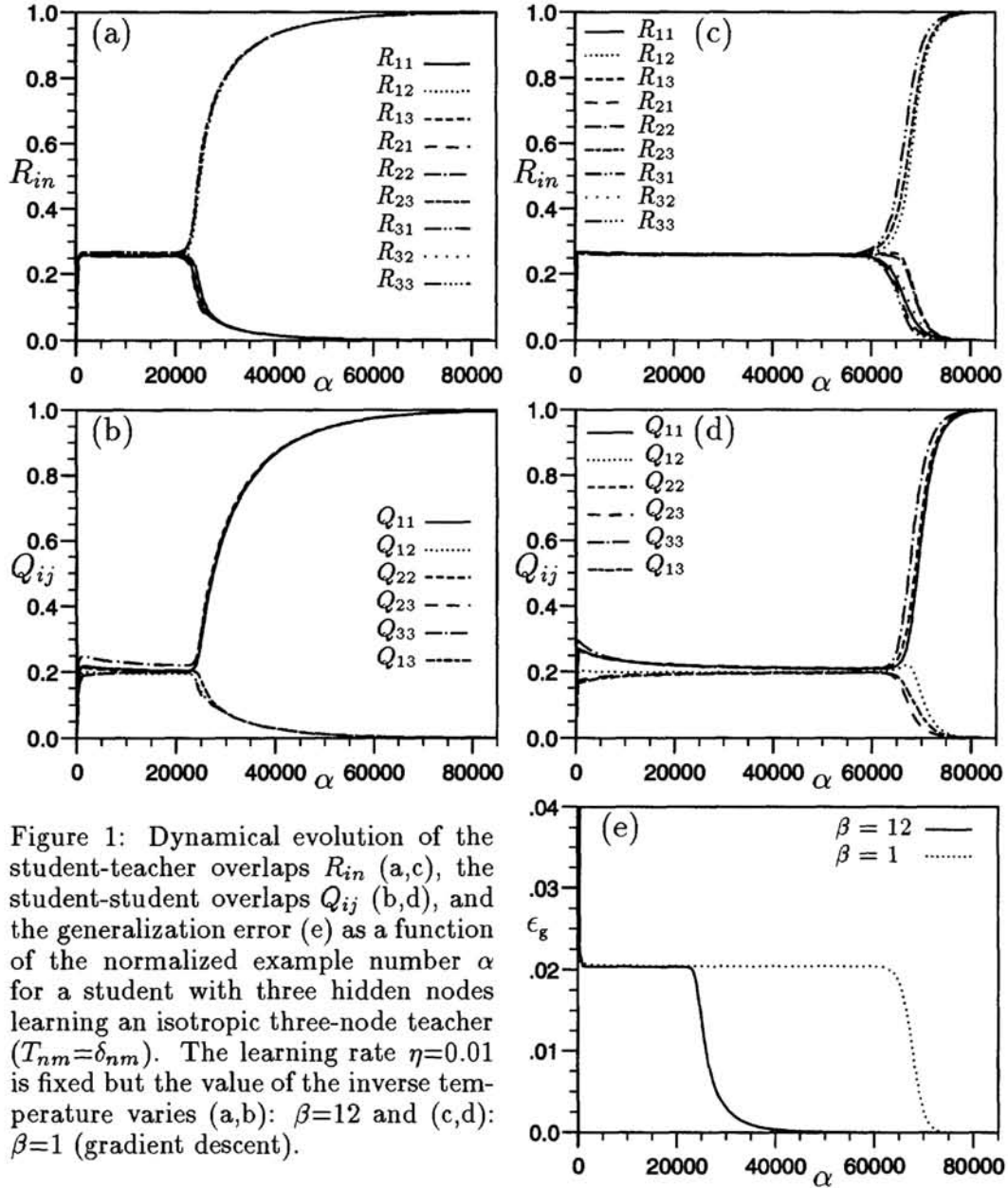

Figure 1: Dynamical evolution of the student-teacher overlaps $R_{in}$ (a,c), the student-student overlaps $Q_{ij}$ (b,d), and the generalization error (e) as a function of the normalized example number $\alpha$ for a student with three hidden nodes learning an isotropic three-node teacher ($T_{nm}=\delta_{nm}$). The learning rate $\eta=0.01$ is fixed but the value of the inverse temperature varies (a,b): $\beta=12$ and (c,d): $\beta=1$ (gradient descent).

## 3 ANALYSIS OF THE DYNAMICAL EQUATIONS

In the case of a realizable learning scenario ($K=M$) and isotropic teachers ($T_{nm}=\delta_{nm}$) the order parameter space can be very well characterized by similar diagonal and off-diagonal elements of the overlap matrices $\mathbf{Q}$ and $\mathbf{R}$, i.e., $Q_{ij} = Q\delta_{ij} + C(1 - \delta_{ij})$ for the student-student overlaps and, apart from a relabeling of the student nodes, by $R_{in} = R\delta_{in} + S(1 - \delta_{in})$ for the student-teacher overlaps. As one can see from Fig. 1, this approximation is particularly good in the symmetric phase and during the final convergence to perfect generalization.

### 3.1 SYMMETRIC PHASE AND ONSET OF SPECIALIZATION

Numerical integration of the equations of motion for a range of learning scenarios show that the length of the symmetric phase is especially prolonged by isotropic teachers and small learning rates $\eta$. We will therefore optimize the dynamics (4) in

the symmetric phase with respect to $\beta$ for isotropic teachers in the small $\eta$ regime, where terms proportional to $\eta^2$ can be neglected. The fixed point of the truncated equations of motion

$$Q^* = C^* = \frac{1}{2K-1} \quad \text{and} \quad R^* = S^* = \sqrt{\frac{Q^*}{K}} = \frac{1}{\sqrt{K(2K-1)}} \quad (5)$$

is independent of $\beta$ and thus identical to the one obtained in [2]. However, the symmetric solution is an unstable fixed point of the dynamics and the small perturbations introduced by the generically nonsymmetric initial conditions will eventually drive the student towards specialization.

To study the onset of specialization, we expand the truncated differential equations to first order in the deviations $q = Q - Q^*$, $c = C - C^*$, $r = R - R^*$, and $s = S - S^*$ from the fixed point values (5). The linearized equations of motion take the form $dv/d\alpha = \mathbf{M} \cdot v$, where $v = (q, c, r, s)$ and $\mathbf{M}$ is a $4 \times 4$ matrix whose elements are the first derivatives of the truncated update equations (4) at the fixed point with respect to $v$. Perturbations or *modes* which are proportional to the *eigenvectors* $v_i$ of $\mathbf{M}$ will therefore decrease or increase exponentially depending on whether the corresponding *eigenvalue* $\lambda_i$ is negative or positive. For the onset of specialization only the modes are relevant which are amplified by the dynamics, i.e., the ones with positive eigenvalue. For them we can identify the inverse eigenvalue as a typical escape time $\tau_i$ from the symmetric phase.

We find only one relevant perturbation for $q = c = 0$ and $s = -r/(K-1)$. This can be confirmed by a closer look at Fig. 1. The onset of specialization is signaled by the breaking of the symmetry between the student-teacher overlaps, whereas significant differences from the symmetric fixed point values of the student norms and overlaps occur later. The escape time $\tau$ associated with the above perturbation is

$$\tau(\beta) = \frac{\pi}{2\eta} \frac{\sqrt{2K-1}(2K+\beta)^{3/2}}{K\beta}. \quad (6)$$

Minimization of $\tau$ with respect to $\beta$ yields $\beta^{\text{opt}} = 4K$, i.e., the optimal $\beta$ scales with the number of hidden units, and

$$\tau^{\text{opt}} = \frac{9\pi}{2\eta} \frac{\sqrt{2K-1}}{\sqrt{6K}}. \quad (7)$$

Trapping in the symmetric phase is therefore always inversely proportional to the learning rate $\eta$. In the large $K$ limit it is proportional to the number of hidden nodes $K$ ($\tau \sim 2\pi K/\eta$) for gradient descent, whereas it is independent of $K$ [$\tau \sim 3\sqrt{3}\pi/(2\eta)$] for the optimized adaptive back-propagation algorithm.

## 3.2   CONVERGENCE TO OPTIMAL GENERALIZATION

In order to predict the optimal learning rate $\eta^{\text{opt}}$ and inverse temperature $\beta^{\text{opt}}$ for the convergence, we linearize the full equations of motion (4) around the zero generalization error fixed point $R^* = Q^* = 1$ and $S^* = C^* = 0$. The matrix $\mathbf{M}$ of the resulting system of four coupled linear differential equations in $q = 1 - Q$, $c = C$, $r = 1 - R$, and $s = S$ is very complicated for arbitrary $\beta$, $K$ and $\eta$, and its eigenvalues and eigenvectors can therefore only be analysed numerically.

We illustrate the solution space with $K = 3$ and two $\beta$ values in Fig. 2a. We find that the dynamics decompose into four modes: two slow modes associated with eigenvalues $\lambda_1$ and $\lambda_2$ and two fast modes associated with eigenvalues $\lambda_3$ and $\lambda_4$, which are negative for all learning rates and whose magnitude is significantly larger.

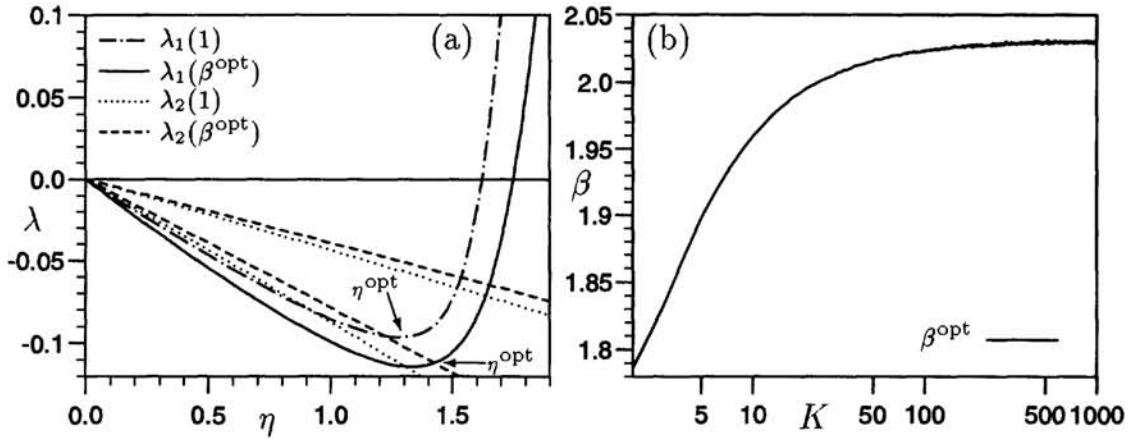

Figure 2: (a) The eigenvalues $\lambda_1, \lambda_2$ (see text) as a function of the learning rate $\eta$ at $K = 3$ for two values of $\beta$: $\beta = 1$, and $\beta = \beta^{\mathrm{opt}} = 1.8314$. For comparison we plot $2\lambda_2$ and find that the optimal learning rate $\eta^{\mathrm{opt}}$ is given by the condition $\lambda_1 = 2\lambda_2$ for $\beta^{\mathrm{opt}}$, but by the minimum of $\lambda_1$ for $\beta = 1$. (b) The optimal inverse temperature $\beta^{\mathrm{opt}}$ as a function of the number of hidden units $K$ saturates for large $K$.

The fast modes decay quickly and their influence on the long-time dynamics is negligible. They are therefore excluded from Fig. 2a and the following discussion. The eigenvalue $\lambda_2$ is negative and linear in $\eta$. The eigenvalue $\lambda_1$ is a non-linear function of both $\beta$ and $\eta$ and negative for small $\eta$. For large $\eta$, $\lambda_1$ becomes positive and training does not converge to the optimal solution defining the maximum learning rate $\eta_{\mathrm{max}}$ as $\lambda_1(\eta_{\mathrm{max}}) = 0$. For all $\eta < \eta_{\mathrm{max}}$ the generalization error decays exponentially to $\epsilon_{\mathrm{g}}{}^* = 0$.

In order to identify the corresponding convergence time $\tau$, which is inversely proportional to the modulus of the eigenvalue associated with the slowest decay mode, we expand the generalization error to second order in $q$, $c$, $r$ and $s$. We find that the mode associated with the linear eigenvalue $\lambda_2$ does not contribute to first order terms, and controls only second order term in a decay rate of $2\lambda_2$. The learning rate $\eta^{\mathrm{opt}}$ which provides the fastest asymptotic decay rate $\lambda^{\mathrm{opt}}$ of the generalization error is therefore either given by the condition $\lambda_1(\eta^{\mathrm{opt}}) = 2\lambda_2(\eta^{\mathrm{opt}})$ or alternatively by $\min_{\eta}(\lambda_1)$ if $\lambda_1 > 2\lambda_2$ at the minimum of $\lambda_1$ (see Fig. 2a).

We can further optimize convergence to optimal generalization by minimizing the decay rate $\lambda^{\mathrm{opt}}(\beta)$ with respect to $\beta$ (see Fig. 2b). Numerically, we find that the optimal inverse temperature $\beta^{\mathrm{opt}}$ saturates for large $K$ at $\beta^{\mathrm{opt}} \approx 2.03$. For large $K$, we find an associated optimal convergence time $\tau^{\mathrm{opt}}(\beta^{\mathrm{opt}}) \sim 2.90K$ for adaptive back-propagation optimized with respect to $\eta$ and $\beta$, which is an improvement by 17% when compared to $\tau^{\mathrm{opt}}(1) \sim 3.48K$ for gradient descent optimized with respect to $\eta$. The optimal and maximal learning rates show an asymptotic $1/K$ behaviour and $\eta^{\mathrm{opt}}(\beta^{\mathrm{opt}}) \sim 4.78/K$, which is an increase by 20% compared to gradient descent. Both algorithms are quite stable as the maximal learning rates, for which the learning process diverges, are about 30% higher than the optimal rates.

## 4  SUMMARY AND DISCUSSION

This research has been motivated by the dominance of the suboptimal symmetric phase in on-line learning of two-layer feedforward networks trained by gradient descent [2]. This trapping is emphasized for inappropriate small learning rates but exists in all training scenarios, effecting the learning process considerably. We

proposed an adaptive back-propagation training algorithm [Eq. (3)] parameterized by an inverse temperature $\beta$, which is designed to improve specialization of the student nodes by enhancing differences in the activation between hidden units. Its performance has been compared to gradient descent for a soft-committee student network with $K$ hidden units trying to learn a rule defined by an isotropic teacher ($T_{nm} = \delta_{nm}$) of the same architecture.

A linear analysis of the equations of motion around the symmetric fixed point for small learning rates has shown that optimized adaptive back-propagation characterized by $\beta^{\text{opt}} = 4K$ breaks the symmetry significantly faster. The effect is especially pronounced for large networks, where the trapping time of gradient descent grows $\tau \propto K/\eta$ compared to $\tau \propto 1/\eta$ for $\beta^{\text{opt}}$. With increasing network size it seems to become harder for a student node trained by gradient descent to distinguish between the many teacher nodes and to specialize on one of them. In the adaptive back-propagation algorithm this effect can be eliminated by choosing $\beta^{\text{opt}} \propto K$.

An open question is how the choice of the optimal inverse temperature is effected for large learning rates, where $\eta^2$-terms cannot be neglected, as unbounded increase of the learning rate causes uncontrolled growth of the student norms. However, the full equations of motion are very difficult to analyse in the symmetric phase. Numerical studies indicate that $\beta^{\text{opt}}$ is smaller but still scales with $K$ and yields an overall decrease in training time which is still significant. We also find that the optimal learning rate $\eta^{\text{opt}}$, which exhibits the shortest symmetric phase, is significantly lower in this regime than during convergence [4].

During convergence, independent of which algorithm is used, the time constant for decay to zero generalization error scales with $K$, due to the necessary rescaling of the learning rate by $1/K$ as the typical quadratic deviation between teacher and student output increases proportional to $K$. The reduction in training time with adaptive back-propagation is 17% and independent of the number of hidden units in contrast to the symmetric phase, where a factor $K$ is gained. This can be explained by the fact that each student node is already specialized on one teacher node and the effect of other nodes in inhibiting further specialization is negligible. In fact, at first it seems rather surprising that anything can be gained by not changing the weights of the network according to their error gradient. The optimal setting of $\beta > 1$, together with training at a larger learning rate, speeds up learning for small activations and slows down learning for highly activated nodes. This is equivalent to favouring rotational changes of the weight vectors over pure length changes to a degree determined by $\beta$.

We believe that the adaptive back-propagation algorithm investigated here will be beneficial for any multilayer feedforward network and hope that this work will motivate further theoretical research into the efficiency of training algorithms and their systematic improvement.

## References

[1]  C. Cybenko, *Math. Control Signals and Systems* **2**, 303 (1989).

[2]  D. Saad and S. A. Solla, *Phys. Rev. E* **52**, 4225 (1995).

[3]  M. Biehl and H. Schwarze, *J. Phys. A* **28**, 643 (1995).

[4]  A. West and D. Saad, in preparation (1995).
